# Supervised Clustering

**Pranjal Awasthi**
Carnegie Mellon University
pawasthi@cs.cmu.edu

**Reza Bosagh Zadeh**
Stanford University
rezab@stanford.edu

## Abstract

Despite the ubiquity of clustering as a tool in unsupervised learning, there is not yet a consensus on a formal theory, and the vast majority of work in this direction has focused on unsupervised clustering. We study a recently proposed framework for supervised clustering where there is access to a teacher. We give an improved generic algorithm to cluster any concept class in that model. Our algorithm is query-efficient in the sense that it involves only a small amount of interaction with the teacher. We also present and study two natural generalizations of the model. The model assumes that the teacher response to the algorithm is perfect. We eliminate this limitation by proposing a noisy model and give an algorithm for clustering the class of intervals in this noisy model. We also propose a dynamic model where the teacher sees a random subset of the points. Finally, for datasets satisfying a spectrum of weak to strong properties, we give query bounds, and show that a class of clustering functions containing Single-Linkage will find the target clustering under the strongest property.

## 1   Introduction

Clustering has traditionally been a tool of unsupervised learning. Despite widespread usage across several fields there is not yet a well-established theory to describe clustering [ABD09, AL10, Blu09, GvLW09]. Recently, Balcan and Blum [BB08] proposed a supervised model of clustering, where there is access to a teacher. We further explore the implications of their model and extend it in several important directions. As a motivating example, consider Google News, where news documents are gathered from the web and need to be clustered into groups, each corresponding to a particular news story. In this case, it is clear to the human eye (the teacher) which group each document should belong to, but the sheer number of articles makes clustering by hand prohibitive. In this case, an algorithm can interact with the teacher to aid in clustering the documents without asking too much of the teacher.

Traditional approaches to clustering optimize some objective function, like the k-means or the k-median, over the given set of points [KVV00, CGTS99]. These approaches work under the implicit assumption that by minimizing a certain objective function one can reach close to the underlying ground truth clustering. Alternatively, another line of work makes strong assumptions on the nature of the data. One popular in literature is the assumption that data is coming from a mixture of Gaussians [Das99]. However when dealing with web-pages, documents etc. it is not very clear if these assumptions are reasonable. In fact, there might be no principled way to reach the target clustering which a teacher has in mind without actually interacting with him/her. For example consider documents representing news articles. These documents could be clustered as {politics, sports, entertainment, other}. However, this is just one of the many possible clusterings. The clustering {entertainment + sports, politics, other} is equally likely apriori. Or perhaps the teacher would like these articles to be clustered into {news articles} vs. {opinion pieces}. These scenarios motivate the need to consider the problem of clustering under feedback. Recently, there has been an interest in investigating such models and to come up with a more formal theoretical framework for analyzing clustering problems and algorithms. One such framework was proposed by Balcan and

Blum [BB08] who, motivated by different models for learning under queries, proposed a model for clustering under queries.

The model is similar to the Equivalence Query(EQ) model of learning [Ang98] but with a different kind of feedback. We assume that the given set $S$ of $m$ points belongs to $k$ target clusters $\{c_1, c_2, \ldots, c_k\}$, where each cluster is defined by some concept $c$ belonging to a concept class $C$. For example, the points belonging to the cluster $c_1$ will be the set $\{x \in S | c_1(x) = 1\}$. We also assume that each point belongs to exactly one of the $k$ clusters. As in the EQ model of learning, the algorithm presents a hypothesis clustering $\{h_1, h_2, \ldots, h_{k'}\}$ to the teacher. If the clustering is incorrect the algorithm gets some feedback from the teacher. However, the feedback in this case is different from the one in the EQ model. In the learning model, the algorithm gets a specific point $x$ as a counter-example to its proposed hypothesis. For clustering problems this may not a very natural form of feedback. In a realistic scenario, the teacher can look at the clustering proposed and give some limited feedback. Hence, the model in [BB08] considers the following feedback: If there is a cluster $h_i$ which contains points from two or more target clusters, then the teacher can ask the algorithm to split that cluster by issuing the request $split(h_i)$. Note that the teacher does not specify how the cluster $h_i$ should be split. If there are clusters $h_i$ and $h_j$ such that $h_i \cup h_j$ is a subset of one of the target clusters, then the teacher can ask the algorithm to merge these two clusters by issuing the request $merge(h_i, h_j)$. The goal of the algorithm is to be query efficient – $O(poly(k, \log m, \log |C|))$ queries, and computationally efficient – running time of $O(poly(k, m, \log |C|))$. Notice, that if we allow the algorithm to use the number of queries linear in $m$, then there is a trivial algorithm, which starts with all the points in separate clusters and then merges clusters as requested by the teacher. One could also imagine applying this split-merge framework to cases where the optimal clustering does not necessarily belong to a natural concept class, but instead satisfies some natural separation conditions (ex., large margin conditions). We also study and present results for such problem instances.

## 1.1   Contributions

In their paper, Balcan and Blum [BB08] gave efficient clustering algorithms for the class of intervals and the class of disjunctions over $\{0, 1\}^n$. We extend those results by constructing an algorithm for clustering the class of axis parallel rectangles in $d$ dimensions. Our algorithm is computationally efficient(for constant $d$) and uses a small number of queries. We generalize our algorithm to cluster the class of hyperplanes in $d$ dimensions with known slopes. Balcan and Blum [BB08] also gave a generic algorithm for any finite concept class $C$, which uses $O(k^3 \log |C|)$ queries. We reduce the query complexity of the generic algorithm from $O(k^3 \log |C|)$ to $O(k \log |C|)$. Furthermore, the new algorithm is much simpler than the one from [BB08]. We study two natural generalization of the original model. In the original model the teacher is only allowed to merge two clusters $h_i$ and $h_j$ if $h_i \cup h_j$ is a subset of one of the target clusters. We consider a noise tolerant version of this in which the teacher can ask the algorithm to merge $h_i$ and $h_j$ if both the clusters have at least some fixed fraction of points belonging to the same target cluster. This is a more natural model since we allow for the teacher requests to be imperfect.

In the original model we assume that the teacher has access to all the points. In practice, we are interested in clustering a large domain of points and the teacher might only have access to a random subset of these points at every step. For example, in the case of clustering news documents, our goal is to figure out the target clustering which reflects the teacher preferences. But the teacher sees a small fresh set of news articles very day. We propose a model which takes into account the fact that at each step the split and merge requests might be on a different set of points. In both the above models the straight forward algorithm for clustering the class of intervals fails. We develop new algorithms for clustering intervals in both the models.

We also apply the split-merge framework of [BB08] to datasets satisfying a spectrum of weak to strong properties and design algorithms for clustering such data sets. Along the way, we also show that a class of clustering functions containing Single-Linkage will find the target clustering under the strict threshold property (Theorem 6.1).

## 2 The model

We consider the model proposed by Balcan and Blum [BB08]. The clustering algorithm is given a set $\mathcal{S}$ of $m$ points. Each point belongs to one of the $k$ clusters. Each cluster is defined by a function $f \in C$, where $C$ is a concept class. The goal of the algorithm is to figure out the correct clustering by interacting with the teacher as follows:

1. The algorithm proposes a hypothesis clustering $\{h_1, h_2, \ldots, h_J\}$ to the teacher.
2. The teacher can request $split(h_i)$ if $h_i$ contains points from two or more target clusters. The teacher can request $merge(h_i, h_j)$ if $h_i \cup h_j$ is a subset of one of the target clusters.

The assumption is that there is no noise in the teacher response. The goal is to use as few queries to the teacher as possible. Ideally, we would like the number of queries to be $poly(k, \log m, \log |C|)$.

### 2.1 A generic algorithm for learning any finite concept class

We reduce the query complexity of the generic algorithm for learning any concept class [BB08], from $O(k^3 \log |C|)$ to $O(k \log |C|)$. In addition our algorithm is simpler than the original one. The new algorithm is described below.

Given $m$ points let $VS = \{$ the set of all possible $k$ clusterings of the given points using concepts in $C\}$. Notice that $|VS| \leq |C|^k$. Given a set $h \subseteq \mathcal{S}$ of points we say that a given clustering $R$ is consistent with $h$ if $h$ appears as a subset of one of the clusters in $R$. Define, $VS(h) = \{R \in VS | R$ is consistent with $h.\}$. At each step the algorithm outputs clusters as follows:

1. Initialize $i = 1$.
2. Find the largest set of points $h_i$, s.t. $|VS(h_i)| \geq \frac{1}{2}|VS|$.
3. Output $h_i$ as a cluster.
4. Set $i = i + 1$ and repeat steps 1-3 on the remaining points until every point has been assigned to some cluster.
5. Present the clustering $\{h_1, h_2, \ldots, h_J\}$ to the teacher.

If the teacher says $split(h_i)$, remove all the clusterings in $VS$ which are consistent with $h_i$ If the teacher says $merge(h_i, h_j)$, remove all the clusterings in $VS$ which are inconsistent with $h_i \cup h_j$.

**Theorem 2.1.** *The generic algorithm can cluster any finite concept class using at most $k \log |C|$ queries.*

*Proof.* At each request, if the teacher says $split(h_i)$, then all the clusterings consistent with $h_i$ are removed, which by the construction followed by the algorithm will be at least half of $|VS|$. If the teacher says $merge(h_i, h_j), i < j$, then all the clusterings inconsistent with $h_i \cup h_j$ are removed. This set will be at least half of $|VS|$, since otherwise the number of clusterings consistent with $h_i \cup h_j$ will be more than half of $|VS|$ which contradicts the maximality of $h_i$. Hence, after each query at least half of the version space is removed. From the above claim we notice that the total number of queries will be at most $\log |VS| \leq log|C|^k \leq k \log |C|$. $\qquad\square$

The analysis can be improved if the VC-dimension $d$ of the concept class $C$ is much smaller than $\log |C|$. In this case the size of $VS$ can be bounded from above by $C[m]^k$, where $C[m]$ is the number of ways to split $m$ points using concepts in $C$. Also from Sauer's lemma[Vap98] we know that $C[m] \leq m^d$. Hence, we get $|VS| \leq m^{kd}$. This gives a query complexity of $O(kd \log m)$.

## 3 Clustering geometric concepts

We now present an algorithm for clustering the class of rectangles in 2 dimensions. We first present a simple but less efficient algorithm for the problem. The algorithm uses $O((k \log m)^3)$ queries and runs in time $poly(k, m)$. In the appendix, we show that the query complexity of the algorithm can be improved to $O((k \log m)^2)$. Our algorithm generalizes in a natural way to rectangles in $d$ dimensional space, and to hyperplanes in $d$ dimensions with known slopes.

### 3.1 An algorithm for clustering rectangles

Each rectangle $c$ in the target clustering can be described by four points $(a_i, a_j), (b_i, b_j)$ such that $(x, y) \in c_k$ iff $a_i < x < a_j$ and $b_i < y < b_j$. Hence, corresponding to any $k$-clustering there are at most $2k$ points $a_1, a_2, \ldots, a_{2k}$ on the $x$-axis and at most $2k$ points $b_1, b_2, \ldots, b_{2k}$ on the $y$-axis. We call these points the *target points*. The algorithm works by finding these points. During its course the algorithm maintains a set of points on the x-axis and a set of points on the y-axis. These points divide the entire space into rectangular regions. The algorithm uses these regions as its hypothesis clusters. The algorithm is sketched below:

1. Start with points $(a_{start}', a_{end}')$ on the x-axis and points $(b_{start}', b_{end}')$, such that all the points are contained in the rectangle defined by these points.

2. At each step, cluster the $m$ points according to the region in which they belong. Present this clustering to the teacher.

3. On a merge request, simply merge the two clusters.

4. On a split of $(a_i', a_j'), (b_i', b_j')$, create a new point $a_r'$ such that $a_i' < a_r' < a_j'$, and the projection of all the points onto $(a_i', a_j')$ is divided into half by $a_r'$. Similarly, create a new point $b_r'$ such that $b_i' < b_r' < b_j'$, and the projection of all the points onto $(b_i', b_j')$ is divided into half by $b_r'$. Abandon all the merges done so far.

**Theorem 3.1.** *The algorithm can cluster the class of rectangles in 2 dimensions using at most* $O((k \log m)^3)$ *queries.*

*Proof.* Lets first bound the total number of split requests. If the teacher says split on $(x_i, x_j), (y_i, y_j)$, then we know that either $(x_i, x_j)$ contains a target point $a$ or $(y_i, y_j)$ contains a target point $b$ or both. By creating two splits we are ensuring that the size of at least one of the regions containing a target point is reduced by half. There are at most $2k$ intervals on the $x$-axis and at most $2k$ intervals on the $y$-axis. Hence, the total number of split requests is $\leq 4k \log m$. Now lets bound the merge requests. Between any two split requests the total number of merge requests will be at most the total number of regions which is $\leq O((k \log m)^2)$. Since, $t$ points on the x and the y axis can create at most $t^2$ regions, we get that the total number of merge requests is at most $\leq O(k \log m)^3$. Hence, the total number of queries made by the algorithm is $O((k \log m)^3)$. $\qquad\square$

If we are a bit more careful, we can avoid redoing the merges after every split and reduce the query complexity to $O((k \log m)^2)$. So, for rectangles we have the following result[1].

**Theorem 3.2.** *There is an algorithm which can cluster the class of rectangles in* 2 *dimensions using at most* $O((k \log m)^2)$ *queries.*

We can also generalize this algorithm to work for rectangles in a $d$-dimensional space. Hence, we get the following result

**Corollary 3.3.** *There is an algorithm which can cluster the class of rectangles in* $d$ *dimensions using at most* $O((kd \log m)^d)$ *queries.*

**Corollary 3.4.** *There is an algorithm which can cluster the class of hyperplanes in* $d$ *dimensions having a known set of slopes of size at most* $s$, *using at most* $O((kds \log m)^d)$ *queries.*

## 4 Dynamic model

We now study a natural generalization of the original model. In the original model we assume that the teacher has access to the entire set of points. In practice, this will rarely be the case. For example, in the case of clustering news articles, each day the teacher sees a small fresh set of articles and provides feedback. Based on this the algorithm must be able to figure out the target clustering for the entire space of articles. More formally, let $X$ be the space of all the points. There is a target $k$ clustering for these points, where cluster corresponds to a concept in a concept class $C$. At each step, the world picks $m$ points and the algorithm clusters these $m$ points and presents the clustering to the teacher. If the teacher is unhappy with the clustering he may provide feedback. Note that

the teacher need not provide feedback every time the algorithm proposes an incorrect clustering. The goal of the algorithm is to minimize the amount of feedback necessary to figure out the target clustering. Notice that at each step the algorithm may get a fresh set of $m$ points. We assume that the requests have no noise and the algorithm has access to all the points in $X$. We now give an algorithm for learning intervals in this model.

## 4.1   An algorithm for clustering intervals

We assume that the space $X$ is discretized into $n$ points. Let us assume that there exist points $\{a_1, a_2, \ldots, a_{k+1}\}$, on the $x$-axis such that the target clustering is the intervals $\{[a_1, a_2], [a_2, a_3], \ldots, [a_k, a_{k+1}]\}$. The algorithm maintains a set of points on the x-axis and uses the intervals induced by them as its hypothesis. Also each interval is associated with a state of $marked/unmarked$. When a new interval is created, it is always $unmarked$. An interval is marked if we know that none of the points($a_i$'s) in the target clustering can be present in that interval. The algorithm is sketched below:

1. Start with one unmarked interval containing all the points in the space.
2. Given a set of $m$ points, first form preliminary clusters $h_1, \ldots, h_J$ such that each cluster corresponds to an interval. Next output the final clusters as follows:
   - set i=1
   - If $h_i$ and $h_{i+1}$ correspond to adjacent intervals at least one of them is unmarked, then output $h_i \cup h_{i+1}$ and set $i = i + 2$. Else output $h_i$ and set $i = i + 1$.
3. On a split request, split every unmarked interval in the cluster in half.
4. On a merge request, mark every unmarked contained in the cluster.

**Theorem 4.1.** *The algorithm can cluster the class of intervals using at most $O(k \log n)$ mistakes.*

*Proof.* Notice that by our construction, every cluster will contain at most 2 unmarked intervals. Lets first bound the total number of split requests. For every point $a_i$ in the target clustering we define two variables $left\_size(a_i)$ and $right\_size(a_i)$. If $a_i$ is inside a hypothesis interval $[x, y]$ then $left\_size(a_i) =$ number of points in $[x, a_i]$ and $right\_size(a_i) =$ number of points in $[a_i, y]$. If $a_i$ is also a boundary point in the hypothesis clustering ($[x, a_i], [a_i, y]$) then again $left\_size(a_i) =$ number of points in $[x, a_i]$ and $right\_size(a_i) =$ number of points in $[a_i, y]$. Notice, that every split request reduces either the $left\_size$ or the $right\_size$ of some boundary point by half. Since there are at most $k$ boundary points in the target clustering, the total number of split requests is $\leq O(k \log n)$ times. Also note that the number of unmarked intervals is at most $O(k \log n)$ since, unmarked intervals increase only via split requests. On every merge request either an unmarked interval is marked or two marked intervals are merged. Hence, the total number of merge requests is atmost twice the number of unmarked intervals $\leq O(k \log n)$. Hence, the total number of mistakes is $\leq O(k \log n)$. □

Its easy to notice that the generic algorithm for learning any finite concept class in the original model also works in this model. Hence, we can learn any finite concept class in this model using at most $k \log |C|$ queries.

## 5   $\eta$ noise model

The previous two models assume that there is no noise in the teacher requests. This is again an unrealistic assumption since we cannot expect the teacher responses to be perfect. For example, if the algorithm proposes a clustering in which there are two clusters which are almost pure,i.e., a large fraction of the points in both the clusters belong to the same target clusters, then there is a good chance that the teacher will ask the algorithm to merge these two clusters, especially if the teacher has access to the clusters through a random subset of the points. In this section we study a model which removes this assumption. For simplicity, we consider the noisy version of the original model [BB08]. As in the original model, the algorithm has $m$ points. At each step, the algorithm proposes a clustering $\{h_1, h_2, \ldots, h_J\}$ to the teacher and the teacher provides feedback. But now, the feedback is noisy in the following sense

1. **Split:** As before the teacher can say $split(h_i)$, if $h_i$ contains points from more than one target clusters.
2. **Merge:** The teacher can say $merge(h_i, h_j)$, if $h_i$ and $h_j$ each have at least one point from some target cluster.

It turns out that handling arbitrary noise is difficult. The following Theorem (proof omitted) shows a counter-example.

**Theorem 5.1.** *Consider $m$ points on a line and $k = 2$. Any clustering algorithm must use $\Omega(m)$ queries in the worst case to figure out the target clustering in the noisy model.*

Hence, we now consider a relaxed notion of noise. If the teacher says $merge(h_i, h_j)$ then we assume that at least a constant $\eta$ fraction of the points in both the clusters, belong to a single target cluster. Under this model of noise we now give an algorithm for learning $k$-intervals.

### 5.1   An algorithm for clustering intervals

The algorithm is a generalization of the interval learning algorithm in the original model. The main idea is that when the teacher asks to merge two intervals $(a_i, a_j)$ and $(a_j, a_k)$, then we know than at least $\eta$ fraction of the portion to the left and the right of $a_j$ is pure. Hence, the algorithm can still make progress. As the algorithm proceeds it is going to mark certain intervals as "pure" which means that all the points in that interval belong to the same cluster. More formally the algorithm is as follows

1. Start with one interval $[a_{start}', a_{end}']$ containing all the points.
2. At each step, cluster the points using the current set of intervals and present that clustering to the teacher.
3. On split request : Divide the interval in half.
4. On a merge request
   - If both the intervals are marked "pure", merge them.
   - If both the intervals are unmarked, then create 3 intervals where the middle interval contains $\eta$ fraction of the two intervals. Also make the middle interval as "pure".
   - If one interval is marked and one is unmarked, then shift the boundary between the two intervals towards the unmarked interval by a fraction of $\eta$.

**Theorem 5.2.** *The algorithm clusters the class of intervals using at most $O(k(\log_{\frac{1}{1-\eta}} m)^2)$.*

*Proof.* We will call a merge request, as "impure" if it involves at least one impure interval,i.e., an interval which contains points from two or more clusters. Else we will call it as "pure". Notice that every split and impure merge request makes progress, i.e. the size of some target interval is reduced by at least $\eta$. Hence, the total number of split + impure merge requests $\leq k \log_{\frac{1}{1-\eta}} m$. We also know that the total number of unmarked intervals $\leq k \log_{\frac{1}{1-\eta}} m$, since only split requests increase the unmarked intervals. Also, total number of marked intervals $\leq$ total number of unmarked intervals, since every marked interval can be charged to a split request. Hence, the total number of intervals $\leq 2k \log_{\frac{1}{1-\eta}} m$.

To bound the total number of pure merges, notice that every time a pure merge is made, the size of some interval decreases by at least an $\eta$ fraction. The size of an interval can decrease at most $\log_{\frac{1}{1-\eta}} m$ times. Hence, the total number of pure merges $\leq k(\log_{\frac{1}{1-\eta}} m)^2$.

Hence, the algorithm makes at most $O(k(\log_{\frac{1}{1-\eta}} m)^2)$ queries.   $\square$

## 6   Properties of the Data

We now adapt the query framework of [BB08] to cluster datasets which satisfy certain natural separation conditions with respect to the target partitioning. For this section, sometimes we write $d = \langle e_1, e_2, \ldots, e_{\binom{n}{2}} \rangle$ to mean the set of distances that exist between all pairs of $n$ points. This

list is *always ordered* by increasing distance. For a definition of the Single-Linkage and Min-Sum clustering functions, please see the appendix.

## 6.1 Threshold Separation

We introduce a (strong) property that may be satisfied by $d = \langle e_1, e_2, \ldots, e_{\binom{n}{2}} \rangle$ with respect to $\Gamma$, the target clustering. It is important to note that this property is imposing restrictions on $d$, defined by the data. An inner edge of $\Gamma$ is a distance between two points inside a cluster, while an outer edge is a distance between two points in differing clusters.

> STRICT THRESHOLD SEPARATION. There exists a threshold $t > 0$ such that all inner edges of $\Gamma$ have distance less than or equal $t$, and all outer edges have distance greater than $t$.

In other words, the pairwise distances between the data are such that all inner edges of $d$ (w.r.t. $\Gamma$) have distance smaller than all outer edges (again, w.r.t. $\Gamma$). This property gives away a lot of information about $\Gamma$, in that it allows Single-Linkage to fully recover $\Gamma$ as we will see in theorem 6.1. Before we present the algorithm to interact with the teacher, Theorem 6.1 will be useful (proof omitted).

[Kle03, JS71] introduce the following 3 properties which a clustering function can satisfy. An $F(d, k)$-transformation of $d$ is a change to $d$ such that inner-cluster distances in $d$ are decreased, and outer-cluster distances are increased.

1. CONSISTENCY. *Fix $k$. Let $d$ be a distance function, and $d'$ be a $F(d, k)$-transformation of $d$. Then $F(d, k) = F(d', k)$*
2. ORDER-CONSISTENCY. *For any two distance functions $d$ and $d'$, number of clusters $k$, if the order of edges in $d$ is the same as the order of edges in $d'$, then $F(d, k) = F(d', k)$*
3. $k$-RICHNESS. *For any number of clusters $k$, Range($F(\bullet, k)$) is equal to the set of all $k$-partitions of $S$*

**Theorem 6.1.** *Fix $k$ and a target $k$-partitioning $\Gamma$, and let $d$ be a distance function satisfying Strict Threshold Separation w.r.t. $\Gamma$. Then for any Consistent, $k$-Rich, Order-Consistent partitioning function $F$, we have $F(d, k) = \Gamma$.*

Note that since Single-linkage is Consistent, $k$-Rich, and Order-Consistent [ZBD09], it immediately follows that $SL(d, k) = \Gamma$ - in other words, SL is guaranteed to find the target $k$-partitioning, but we still have to interact with the teacher to find out $k$. It is a recently resolved problem that Single-Linkage is not the only function satisfying the above properties [ZBD], so the the class of Consistent, $k$-Rich, and Order-Consistent functions has many members. We now present the algorithm to interact with the teacher.

**Theorem 6.2.** *Given a dataset satisfying Strict Threshold Separation, there exists an algorithm which can find the target partitioning for any hypothesis class in $O(\log(n))$ queries*

*Proof.* Note that the threshold $t$ and the number of clusters $k$ are not known to the algorithm, else the target could be found immediately. By theorem 6.1, we know that the target must be exactly what Single-Linkage returns for some $k$, and it remains to find the number of clusters. This can be done using a binary search on the number of clusters which can vary from 1 to $n$. We start with some candidate $k$, and if the teacher tells us to split anything, we know the number of clusters must be larger, and if we are told to merge, we know the number of clusters must be smaller. Thus we can find the correct number of clusters in $O(\log(n))$ queries. □

Note that since strict threshold separation implies strict separation, then the $O(k)$ algorithm presented in the next section can also be used, giving $O(\min(\log(n), k))$ queries.

**Strict Separation:** Now we relax strict threshold separation

> STRICT SEPARATION. All points in the same cluster are more similar to one another than to points outside the cluster.

With this property, it is no longer true that all inner distances are smaller than outer distances, and therefore Theorem 6.1 does not apply. However, [BBV08] prove the following lemma

**Lemma 6.3.** *[BBV08] For a dataset satisfying strict separation, let $SL(d)$ be the tree returned by Single-Linkage. Then any partitioning respecting the strict separation of $d$ will be a pruning of $SL(d)$.*

**Theorem 6.4.** *Given a dataset satisfying Strict Separation, there exists an algorithm which can find the target partitioning for any hypothesis class in $O(k)$ queries*

*Proof.* Let the distances between points be represented by the distance function $d$. By lemma 6.3 we know that the target partitioning must be a pruning of $SL(d)$. Our algorithm will start by presenting the teacher with all points in a single cluster. Upon a split request, we split according to the relevant node in $SL(d)$. There can be no merge requests since we always split perfectly. Each split will create a new cluster, so there will be at most $k-1$ of these splits, after which the correct partitioning is found. □

$\gamma$**-margin Separation:** Margins show up in many learning models, and this is no exception. A natural assumption is that there may be a separation of at least $\gamma$ between points in differing clusters, where the points all lie inside the unit ball.

$\gamma$-MARGIN SEPARATION. Points in different clusters of the target partitioning are at least $\gamma$ away from one another.

With this property, we can prove the following for all hypothesis classes

**Theorem 6.5.** *Given a dataset satisfying $\gamma$-margin Separation, there exists an algorithm which can find the target partitioning for any hypothesis class in $O((\frac{\sqrt{d}}{\gamma})^d - k)$ queries*

*Proof.* We split the unit ball (inside which all points live) into hypercubes with edge length $\frac{\gamma}{\sqrt{d}}$. We are interested in the diameter of such a hypercube. The diameter of a $d$-dimensional hypercube with side $\frac{\gamma}{\sqrt{d}}$ is $\sqrt{d} \times \frac{\gamma}{\sqrt{d}} = \gamma$, so no two points inside a hypercube of side $\frac{\gamma}{\sqrt{d}}$ can be more than $\gamma$ apart. It follows that if split the unit ball up using a grid of hypercubes, all points inside a hypercube must be from the same cluster. We say such a hypercube is "pure".

There are at most $O((\frac{\sqrt{d}}{\gamma})^d)$ hypercubes in a unit ball. We show each hypercube as a single cluster to the teacher. Since all hypercubes are pure, we can only get merge requests, of which there can be at most $O((\frac{\sqrt{d}}{\gamma})^d - k)$ until the target partitioning is found. □

# 7 Conclusions and open problems

In this paper we investigated a recently proposed model of clustering under feedback. We gave algorithms for clustering geometric concepts in the model. For datasets satisfying a spectrum of weak to strong properties, we gave query bounds, and showed that a class of clustering functions containing Single-Linkage will find the target clustering under the strongest property. We also studied natural generalizations of the model and gave efficient algorithms for learning intervals in the new models. Several interesting problems remain

1. Give algorithms for clustering other classes of functions, for example linear separators in the original model.

2. Give efficient algorithms for clustering geometric concept classes in the new models.

3. Establish connections between the proposed models and the Equivalence Query model of learning.

4. In [BB08], the authors give an algorithm for learning the class of disjunctions. It would be interesting to come up with an attribute efficient version of the algorithm, similar in spirit to the Winnow algorithm [Lit87].

## Footnotes

[1]Proof is omitted due to space constraints

# References

[ABD09]  M. Ackerman and S. Ben-David. Clusterability: A theoretical study. *Proceedings of AISTATS-09, JMLR: W&CP*, 5:1–8, 2009.

[AL10]  Ben-David S. Ackerman, M. and D. Loker. Characterization of Linkage-based Clustering. *COLT 2010*, 2010.

[Ang98]  D. Angluin. Queries and concept learning. *Machine Learning*, 2:319–342, 1998.

[BB08]  Maria-Florina Balcan and Avrim Blum. Clustering with interactive feedback. In *ALT*, 2008.

[BBV08]  M.-F. Balcan, A. Blum, and S. Vempala. A discriminative framework for clustering via similarity functions. In *Proceedings of the 40th ACM Symposium on Theory of Computing*, 2008.

[Blu09]  Avrim Blum. Thoughts on clustering. In *NIPS Workshop on Clustering Theory*, 2009.

[CGTS99]  M. Charikar, S. Guha, E. Tardos, and D. B. Shmoy. A constant-factor approximation algorithm for the k-median problem. In *ACM Symposium on Theory of Computing*, 1999.

[Das99]  S. Dasgupta. Learning mixtures of gaussians. In *Proceedings of the 40th Annual Symposium on Foundations of Computer Science*, 1999.

[GvLW09]  I. Guyon, U. von Luxburg, and R.C. Williamson. Clustering: Science or Art? In *NIPS Workshop on Clustering Theory*, 2009.

[JS71]  N. Jardine and R. Sibson. Mathematical taxonomy. *New York*, 1971.

[Kle03]  J. Kleinberg. An impossibility theorem for clustering. In *Advances in Neural Information Processing Systems 15: Proceedings of the 2002 Conference*, page 463. The MIT Press, 2003.

[KVV00]  R. Kannan, S. Vempala, and A. Veta. On clusterings-good, bad and spectral. In *FOCS '00: Proceedings of the 41st Annual Symposium on Foundations of Computer Science*, 2000.

[Lit87]  Nick Littlestone. Learning quickly when irrelevant attributes abound: A new linear-threshold algorithm. *Machine Learning*, 2(4), 1987.

[Vap98]  V. N. Vapnik. *Statistical Learning Theory*. John Wiley and Sons Inc., 1998.

[ZBD]  Reza Bosagh Zadeh and Shai Ben-David. Axiomatic Characterizations of Single-Linkage. In *In Submission*.

[ZBD09]  Reza Bosagh Zadeh and Shai Ben-David. A Uniqueness Theorem for Clustering. In *Proceedings of the 25th Conference on Uncertainty in Artificial Intelligence*, 2009.

